# Predicting Speech Intelligibility from a Population of Neurons

**Jeff Bondy**
Dept. of Electrical Engineering
McMaster University
Hamilton, ON
*jeff@soma.crl.mcmaster.ca*

**Ian C. Bruce**
Dept. of Electrical Engineering
McMaster University
Hamilton, ON
*ibruce@ieee.org*

**Suzanna Becker**
Dept. of Psychology
McMaster University
*becker@mcmaster.ca*

**Simon Haykin**
Dept. of Electrical Engineering
McMaster University
*haykin@mcmaster.ca*

## Abstract

A major issue in evaluating speech enhancement and hearing compensation algorithms is to come up with a suitable metric that predicts intelligibility as judged by a human listener. Previous methods such as the widely used Speech Transmission Index (STI) fail to account for masking effects that arise from the highly nonlinear cochlear transfer function. We therefore propose a Neural Articulation Index (NAI) that estimates speech intelligibility from the instantaneous neural spike rate over time, produced when a signal is processed by an auditory neural model. By using a well developed model of the auditory periphery and detection theory we show that human perceptual discrimination closely matches the modeled distortion in the instantaneous spike rates of the auditory nerve. In highly rippled frequency transfer conditions the NAI's prediction error is 8% versus the STI's prediction error of 10.8%.

## 1  Introduction

A wide range of intelligibility measures in current use rest on the assumption that intelligibility of a speech signal is based upon the sum of contributions of intelligibility within individual frequency bands, as first proposed by French and Steinberg [1]. This basic method applies a function of the Signal-to-Noise Ratio (SNR) in a set of bands, then averages across these bands to come up with a prediction of intelligibility. French and Steinberg's original Articulation Index (AI) is based on 20 equally contributing bands, and produces an intelligibility score between zero and one:

$$AI = \frac{1}{20} \sum_{i=1}^{20} TI_i \quad , \tag{1}$$

where $TI_i$ (Transmission Index i) is the normalized intelligibility in the $i^{th}$ band. The TI per band is a function of the signal to noise ratio or:

$$TI_i = \frac{SNR_i + 12}{30} \qquad (2)$$

for SNRs between –12 dB and 18 dB. A SNR of greater than 18 dB means that the band has perfect intelligibility and TI equals 1, while an SNR under –12 dB means that a band is not contributing at all, and the TI of that band equals 0. The overall intelligibility is then a function of the AI, but this function changes depending on the semantic context of the signal.

Kryter validated many of the underlying AI principles [2]. Kryter also presented the mechanics for calculating the AI for different number of bands - 5,6,15 or the original 20 - as well as important correction factors [3]. Some of the most important correction factors account for the effects of modulated noise, peak clipping, and reverberation. Even with the application of various correction factors, the AI does not predict intelligibility in the presence of some time-domain distortions. Consequently, the Modulation Transfer Function (MTF) has been utilized to measure the loss of intelligibility due to echoes and reverberation [4]. Steeneken and Houtgast later extended this approach to include nonlinear distortions, giving a new name to the predictor: the Speech Transmission Index (STI) [5]. These metrics proved more valid for a larger range of environments and interferences.

The STI test signal is a long-term average speech spectrum, gaussian random signal, amplitude modulated by a 0.63 Hz to 12.5 Hz tone. Acoustic components within different frequency bands are switched on and off over the testing sequence to come up with an intelligibility score between zero and one. Interband intermodulation sources can be discerned, as long as the product does not fall into the testing band. Therefore, the STI allows for standard AI-frequency band weighted SNR effects, MTF-time domain effects, and some limited measurements of nonlinearities. The STI shows a high correlation with empirical tests, and has been codified as an ANSI standard [6]. For general acoustics it is very good. However, the STI does not accurately model intraband masker non-linearities, phase distortions or the underlying auditory mechanisms (outside of independent frequency bands)

We therefore sought to extend the AI/STI concepts to predict intelligibility, on the assumption that the closest physical variable we have to the perceptual variable of intelligibility is the auditory nerve response. Using a spiking model of the auditory periphery [7] we form the Neuronal Articulation Index (NAI) by describing distortions in the spike trains of different frequency bands. The spiking over time of an auditory nerve fiber for an undistorted speech signal (control case) is compared to the neural spiking over time for the same signal after undergoing some distortion (test case). The difference in the estimated instantaneous discharge rate for the two cases is used to calculate a neural equivalent to the TI, the Neural Distortion (ND), for each frequency band. Then the NAI is calculated with a weighted average of NDs at different Best Frequencies (BFs). In general detection theory terms, the control neuronal response sets some locus in a high dimensional space, then the distorted neuronal response will project near that locus if it is perceptually equivalent, or very far away if it is not. Thus, the distance between the control neuronal response and the distorted neuronal response is a function of intelligibility. Due to the limitations of the STI mentioned above it is predicted that a measure of the neural coding error will be a better predictor than SNR for human intelligibility word-scores. Our method also has the potential to shed light on the underlying neurobiological mechanisms.

## 2 Method

### 2.1 Model

The auditory periphery model used throughout (and hereafter referred to as the Auditory Model) is from [7]. The system is shown in Figure 1.

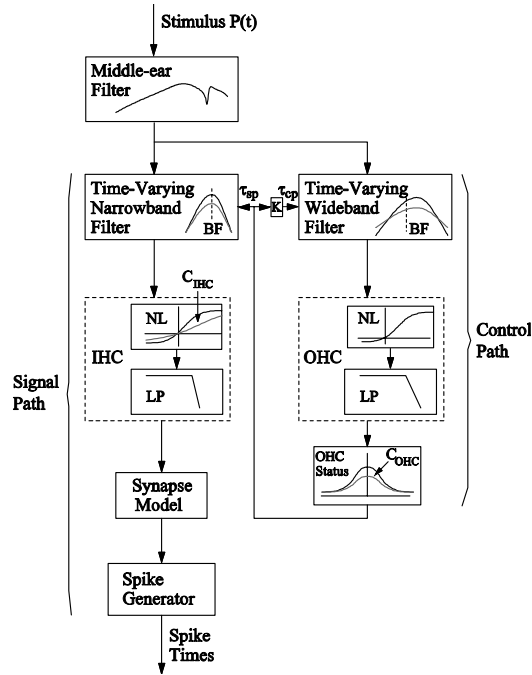

**Figure 1 Block diagram of the computational model of the auditory periphery from the middle ear to the Auditory Nerve. Reprinted from Fig. 1 of [7] with permission from the Acoustical Society of America © (2003).**

The auditory periphery model comprises several sections, each providing a phenomenological description of a different part of the cat auditory periphery function.

The first section models middle ear filtering. The second section, labeled the "control path," captures the Outer Hair Cells (OHC) modulatory function, and includes a wideband, nonlinear, time varying, band-pass filter followed by an OHC nonlinearity (NL) and low-pass (LP) filter. This section controls the time-varying, nonlinear behavior of the narrowband signal-path basilar membrane (BM) filter. The control-path filter has a wider bandwidth than the signal-path filter to account for wideband nonlinear phenomena such as two-tone rate suppression.

The third section of the model, labeled the "signal path", describes the filter properties and traveling wave delay of the BM (time-varying, narrowband filter); the nonlinear transduction and low-pass filtering of the Inner Hair Cell (IHC NL and LP); spontaneous and driven activity and adaptation in synaptic transmission (synapse model); and spike generation and refractoriness in the auditory nerve (AN). In this model, $C_{IHC}$ and $C_{OHC}$ are scaling constants that control IHC and OHC status, respectively.

The parameters of the synapse section of the model are set to produce adaptation and discharge-rate versus level behavior appropriate for a high-spontaneous-

rate/low-threshold auditory nerve fiber. In order to avoid having to generate many spike trains to obtain a reliable estimate of the instantaneous discharge rate over time, we instead use the synaptic release rate as an approximation of the discharge rate, ignoring the effects of neural refractoriness.

## 2.2 Neural articulation index

These results emulate most of the simulations described in Chapter 2 of Steeneken's thesis [8], as it describes the full development of an STI metric from inception to end. For those interested, the following simulations try to map most of the second chapter, but instead of basing the distortion metric on a SNR calculation, we use the neural distortion.

There are two sets of experiments. The first, in section 3.1, deals with applying a frequency weighting structure to combine the band distortion values, while section 3.2 introduces redundancy factors also. The bands, chosen to match [8], are octave bands centered at [125, 250, 500, 1000, 2000, 4000, 8000] Hz. Only seven bands are used here. The Neural AI (NAI) for this is:

$$NAI = \alpha_1 \cdot NTI_1 + \alpha_2 \cdot NTI_2 + ... + \alpha_7 \cdot NTI_7 \ , \qquad (3)$$

where $\bullet_i$ is the $i^{th}$ bands contribution and $NTI_i$ is the Neural Transmission Index in the $i^{th}$ band. Here all the $\bullet$s sum to one, so each $\bullet$ factor can be thought of as the percentage contribution of a band to intelligibility. Since NTI is between [0,1], it can also be thought of as the percentage of acoustic features that are intelligible in a particular band. The ND per band is the projection of the distorted (Test) instantaneous spike rate against the clean (Control) instantaneous spike rate.

$$ND = 1 - \frac{Test \cdot Control^T}{Control \cdot Control^T} \ , \qquad (4)$$

where Control and Test are vectors of the instantaneous spike rate over time, sampled at 22050 Hz. This type of error metric can only deal with steady state channel distortions, such as the ones used in [8]. ND was then linearly fit to resemble the TI equation 1-2, after normalizing each of the seven bands to have zero means and unit standard deviations across each of the seven bands. The NTI in the $i^{th}$ band was calculated as

$$NTI_i = m\frac{ND_i - \mu_i}{\sigma_i} + b \ . \qquad (5)$$

$NTI_i$ is then thresholded to be no less then 0 and no greater then 1, following the TI thresholding. In equation (5) the factors, $m = 2.5$, $b = -1$, were the best linear fit to produce $NTI_i$'s in bands with SNR greater then 15 dB of 1, bands with 7.5 dB SNR produce $NTI_i$'s of 0.75, and bands with 0 dB SNR produced $NTI_i$'s of 0.5. This closely followed the procedure outlined in section 2.3.3 of [8]. As the TI is a best linear fit of SNR to intelligibility, the NTI is a best linear fit of neural distortion to intelligibility.

The input stimuli were taken from a Dutch corpus [9], and consisted of 10 Consonant-Vowel-Consonant (CVC) words, each spoken by four males and four females and sampled at 44100 Hz. The Steeneken study had many more, but the exact corpus could not be found. 80 total words is enough to produce meaningful frequency weighting factors. There were 26 frequency channel distortion conditions used for male speakers, 17 for female and three SNRs (+15 dB, +7.5 dB and 0 dB). The channel conditions were split into four groups given in Tables 1 through 4 for males, since females have negligible signal in the 125 Hz band, they used a subset, marked with an asterisk in Table 1 through Table 4.

Table 1: Rippled Envelope

| ID # | OCTAVE-BAND CENTRE FREQUENCY | | | | | | |
|---|---|---|---|---|---|---|---|
| | 125 | 250 | 500 | 1K | 2K | 4K | 8K |
| 1* | 1 | 1 | 1 | 1 | 0 | 0 | 0 |
| 2* | 0 | 0 | 0 | 0 | 1 | 1 | 1 |
| 3* | 1 | 1 | 0 | 0 | 0 | 1 | 1 |
| 4* | 0 | 0 | 1 | 1 | 1 | 0 | 0 |
| 5* | 1 | 1 | 0 | 0 | 1 | 1 | 0 |
| 6* | 0 | 0 | 1 | 1 | 0 | 0 | 1 |
| 7* | 1 | 0 | 1 | 0 | 1 | 0 | 1 |
| 8* | 0 | 1 | 0 | 1 | 0 | 1 | 0 |

Table 2: Adjacent Triplets

| ID # | OCTAVE-BAND CENTRE FREQUENCY | | | | | | |
|---|---|---|---|---|---|---|---|
| | 125 | 250 | 500 | 1K | 2K | 4K | 8K |
| 9 | 1 | 1 | 1 | 0 | 0 | 0 | 0 |
| 10 | 0 | 1 | 1 | 1 | 0 | 0 | 0 |
| 11* | 0 | 0 | 0 | 1 | 1 | 1 | 0 |

Table 3: Isolated Triplets

| ID # | OCTAVE-BAND CENTRE FREQUENCY | | | | | | |
|---|---|---|---|---|---|---|---|
| | 125 | 250 | 500 | 1K | 2K | 4K | 8K |
| 12 | 1 | 0 | 1 | 0 | 1 | 0 | 0 |
| 13 | 1 | 0 | 1 | 0 | 0 | 1 | 0 |
| 14 | 1 | 0 | 0 | 1 | 0 | 1 | 0 |
| 15* | 0 | 1 | 0 | 1 | 0 | 0 | 1 |
| 16* | 0 | 1 | 0 | 0 | 1 | 0 | 1 |
| 17 | 0 | 0 | 1 | 0 | 1 | 0 | 1 |

Table 4: Contiguous Bands

| ID # | OCTAVE-BAND CENTRE FREQUENCY | | | | | | |
|---|---|---|---|---|---|---|---|
| | 125 | 250 | 500 | 1K | 2K | 4K | 8K |
| 18* | 0 | 1 | 1 | 1 | 1 | 0 | 0 |
| 19* | 0 | 0 | 1 | 1 | 1 | 1 | 0 |
| 20* | 0 | 0 | 0 | 1 | 1 | 1 | 1 |
| 21 | 1 | 1 | 1 | 1 | 1 | 0 | 0 |
| 22* | 0 | 1 | 1 | 1 | 1 | 1 | 0 |
| 23* | 0 | 0 | 1 | 1 | 1 | 1 | 1 |
| 24 | 1 | 1 | 1 | 1 | 1 | 1 | 0 |
| 25 | 0 | 1 | 1 | 1 | 1 | 1 | 1 |
| 26* | 1 | 1 | 1 | 1 | 1 | 1 | 1 |

In the above tables a one represents a passband and a zero a stop band. A 1353 tap FIR filter was designed for each envelope condition. The female envelopes are a subset of these because they have no appreciable speech energy in the 125 Hz octave band. Using the 40 male utterances and 40 female utterances under distortion and calculating the NAI following equation (3) produces only a value between [0,1]. To produce a word-score intelligibility prediction between zero and 100 percent the NAI value was fit to a third order polynomial that produced the lowest standard deviation of error from empirical data. While Fletcher and Galt [10] state that the relation between AI and intelligibility is exponential, [8] fits with a third order polynomial, and we have chosen to compare to [8]. The empirical word-score intelligibility was from [8].

# 3 Results

## 3.1 Determining frequency weighting structure

For the first tests, the optimal frequency weights (the values of $\bullet_i$ from equation 3) were designed through minimizing the difference between the predicted intelligibility and the empirical intelligibility. At each iteration one of the values was dithered up or down, and then the sum of the $\bullet_i$ was normalized to one. This is very similar to [5] whose final standard deviation of prediction error for males was 12.8%, and 8.8% for females. The NAI's final standard deviation of prediction error for males was 8.9%, and 7.1% for females.

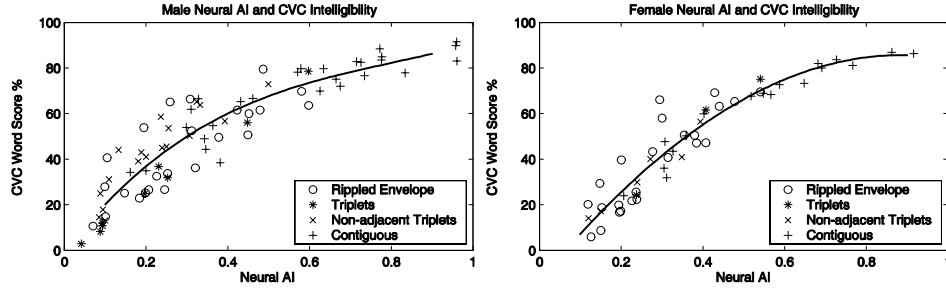

**Figure 2 Relation between NAI and empirical word-score intelligibility for male (left) and female (right) speech with bandpass limiting and noise. The vertical spread from the best fitting polynomial for males has a s.d. = 8.9% versus the STI [5] s.d. = 12.8%, for females the fit has a s.d. = 7.1% versus the STI [5] s.d. = 8.8%**

The frequency weighting factors are similar for the NAI and the STI. The STI weighting factors from [8], which produced the optimal prediction of empirical data (male s.d. = 6.8%, female s.d. = 6.0%) and the NAI are plotted in Figure 3.

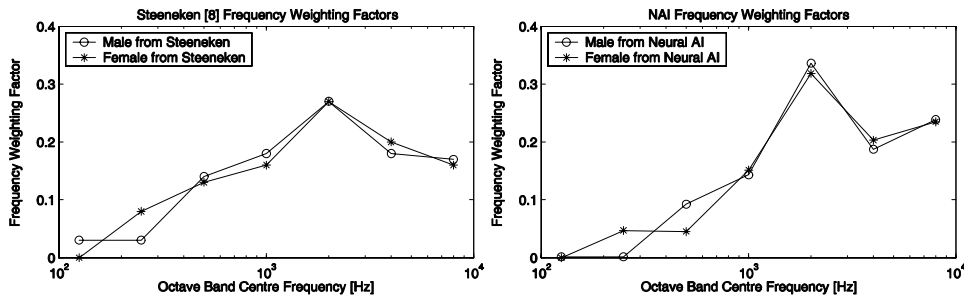

**Figure 3 Frequency weighting factors for the optimal predictor of male and female intelligibility calculated with the NAI and published by Steeneken [8].**

As one can see, the low frequency information is tremendously suppressed in the NAI, while the high frequencies are emphasized. This may be an effect of the stimuli corpus. The corpus has a high percentage of stops and fricatives in the initial and final consonant positions. Since these have a comparatively large amount of high frequency signal they may explain this discrepancy at the cost of the low frequency weights. [8] does state that these frequency weights are dependant upon the conditions used for evaluation.

## 3.2 Determining frequency weighting with redundancy factors

In experiment two, rather then using equation (3) that assumes each frequency band contributes independently, we introduce redundancy factors. There is correlation between the different frequency bands of speech [11], which tends to make the STI over-predict intelligibility. The redundancy factors attempt to remove correlate signals between bands. Equation (3) then becomes:

$$NAI_r = \alpha_1 \cdot NTI_1 - \beta_1\sqrt{NTI_1 \cdot NTI_2} + \alpha_2 \cdot NTI_2 - \beta_1\sqrt{NTI_2 \cdot NTI_3} + ... + \alpha_7 \cdot NTI_7 \ ,(6)$$

where the r subscript denotes a redundant NAI and • is the correlation factor. Only adjacent bands are used here to reduce complexity. We replicated Section 3.1 except using equation 6. The same testing, and adaptation strategy from Section 3.1 was used to find the optimal •s and •s.

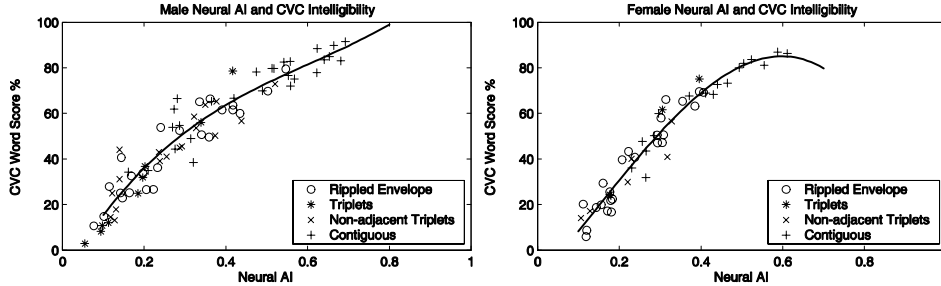

**Figure 4 Relation between NAI$_r$ and empirical word-score intelligibility for male speech (right) and female speech (left) with bandpass limiting and noise with Redundancy Factors. The vertical spread from the best fitting polynomial for males has a s.d. = 6.9% versus the STI$_r$ [8] s.d. = 4.7%, for females the best fitting polynomial has a s.d. = 5.4% versus the STI$_r$ [8] s.d. = 4.0%.**

The frequency weighting and redundancy factors given as optimal in Steeneken, versus calculated through optimizing the NAI$_r$ are given in Figure 5.

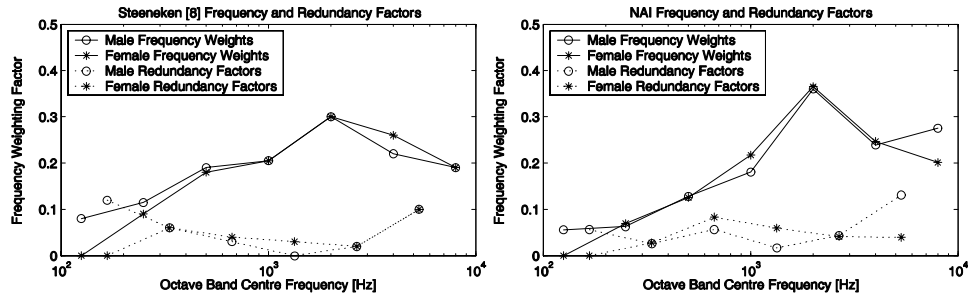

**Figure 5 Frequency and redundancy factors for the optimal predictor of male and female intelligibility calculated with the NAI$_r$ and published in [8].**

The frequency weights for the NAI$_r$ and STI$_r$ are more similar than in Section 3.1. The redundancy factors are very different though. The NAI redundancy factors show no real frequency dependence unlike the convex STI redundancy factors. This may be due to differences in optimization that were not clear in [8].

Table 5: Standard Deviation of Prediction Error

|  | MALE EQ. 3 | FEMALE EQ. 3 | MALE EQ. 6 | FEMALE EQ. 6 |
|---|---|---|---|---|
| **NAI** | 8.9 % | 7.1 % | 6.9 % | 5.4 % |
| **STI [5]** | 12.8 % | 8.8 % |  |  |
| **STI [8]** | 6.8 % | 6.0 % | 4.7 % | 4.0 % |

The mean difference in error between the $STI_r$, as given in [8], and the $NAI_r$ is 1.7%. This difference may be from the limited CVC word choice. It is well within the range of normal speaker variation, about 2%, so we believe that the NAI and $NAI_r$ are comparable to the STI and $STI_r$ in predicting speech intelligibility.

## 4   Conclusions

These results are very encouraging. The NAI provides a modest improvement over STI in predicting intelligibility. We do not propose this as a replacement for the STI for general acoustics since the NAI is much more computationally complex then the STI. The NAI's end applications are in predicting hearing impairment intelligibility and using statistical decision theory to describe the auditory systems feature extractors - tasks which the STI cannot do, but are available to the NAI.

While the AI and STI can take into account threshold shifts in a hearing impaired individual, neither can account for sensorineural, suprathreshold degradations [12]. The accuracy of this model, based on cat anatomy and physiology, in predicting human speech intelligibility provides strong validation of attempts to design hearing aid amplification schemes based on physiological data and models [13]. By quantifying the hearing impairment in an intelligibility metric by way of a damaged auditory model one can provide a more accurate assessment of the distortion, probe how the distortion is changing the neuronal response and provide feedback for preprocessing via a hearing aid before the impairment. The NAI may also give insight into how the ear codes stimuli for the very robust, human auditory system.

## References

[1] French, N.R. & Steinberg, J.C. (1947) Factors governing the intelligibility of speech sounds. *J. Acoust. Soc. Am.* **19**:90-119.

[2] Kryter, K.D. (1962) Validation of the articulation index. *J. Acoust. Soc. Am.* **34**:1698-1702.

[3] Kryter, K.D. (1962b) Methods for the calculation and use of the articulation index. *J. Acoust. Soc. Am.* **34**:1689-1697.

[4] Houtgast, T. & Steeneken, H.J.M. (1973) The modulation transfer function in room acoustics as a predictor of speech intelligibility. *Acustica* **28**:66-73.

[5] Steeneken, H.J.M. & Houtgast, T. (1980) A physical method for measuring speech-transmission quality. *J. Acoust. Soc. Am.* **67**(1):318-326.

[6] ANSI (1997) ANSI S3.5-1997 Methods for calculation of the speech intelligibility index. *American National Standards Institute*, New York.

[7] Bruce, I.C., Sachs, M.B., Young, E.D. (2003) An auditory-periphery model of the effects of acoustic trauma on auditory nerve responses. *J. Acoust. Soc. Am.*, **113**(1):369-388.

[8] Steeneken, H.J.M. (1992) On measuring and predicting speech intelligibility. Ph.D. Dissertation, University of Amsterdam.

[9] van Son, R.J.J.H., Binnenpoorte, D., van den Heuvel, H. & Pols, L.C.W. (2001) The IFA corpus: a phonemically segmented Dutch "open source" speech database. *Eurospeech 2001 Poster* http://145.18.230.99/corpus/index.html

[10] Fletcher, H., & Galt, R.H. (1950) The perception of speech and its relation to telephony. *J. Acoust. Soc. Am.* **22**:89-151.

[11] Houtgast, T., & Verhave, J. (1991) A physical approach to speech quality assessment: correlation patterns in the speech spectrogram. *Proc. Eurospeech 1991*, Genova:285-288.

[12] van Schijndel, N.H., Houtgast, T. & Festen, J.M. (2001) Effects of degradation of intensity, time, or frequency content on speech intelligibility for normal-hearing and hearing-impaired listeners. *J. Acoust. Soc. Am.***110**(1):529-542.

[13] Sachs, M.B., Bruce, I.C., Miller, R.L., & Young, E. D. (2002) Biological basis of hearing-aid design. *Ann. Biomed. Eng.* **30**:157–168.
